# Efficient coding of natural images with a population of noisy Linear-Nonlinear neurons

**Yan Karklin and Eero P. Simoncelli**
Howard Hughes Medical Institute and
Center for Neural Science
New York University
New York, NY 10003
{yan.karklin, eero.simoncelli}@nyu.edu

## Abstract

Efficient coding provides a powerful principle for explaining early sensory coding. Most attempts to test this principle have been limited to linear, noiseless models, and when applied to natural images, have yielded oriented filters consistent with responses in primary visual cortex. Here we show that an efficient coding model that incorporates biologically realistic ingredients – input and output noise, nonlinear response functions, and a metabolic cost on the firing rate – predicts receptive fields and response nonlinearities similar to those observed in the retina. Specifically, we develop numerical methods for simultaneously learning the linear filters and response nonlinearities of a population of model neurons, so as to maximize information transmission subject to metabolic costs. When applied to an ensemble of natural images, the method yields filters that are center-surround and nonlinearities that are rectifying. The filters are organized into two populations, with On- and Off-centers, which independently tile the visual space. As observed in the primate retina, the Off-center neurons are more numerous and have filters with smaller spatial extent. In the absence of noise, our method reduces to a generalized version of independent components analysis, with an adapted nonlinear "contrast" function; in this case, the optimal filters are localized and oriented.

## 1 Introduction

Coding efficiency is a well-known objective for the evaluation and design of signal processing systems, and provides a theoretical framework for understanding biological sensory systems. Attneave [1] and Barlow [2] proposed that early sensory systems are optimized, subject to the limitations of their available resources, for representing information contained in naturally occurring stimuli. Although these proposals originated more than 50 years ago, they have proven difficult to test. The optimality of a given sensory representation depends on the family of possible neural transformations to which it is compared, the costs of building, maintaining, and operating the system, the distribution of input signals over which the system is evaluated, and the levels of noise in the input and output.

A substantial body of work has examined coding efficiency of early visual representations. For example, the receptive fields of retinal neurons have been shown to be consistent with efficient coding principles [3, 4, 5, 6]. However, these formulations rely on unrealistic assumptions of linear response and Gaussian noise, and their predictions are not uniquely constrained. For example, the observation that band-pass filtering is optimal [4] is insufficient to explain rotationally symmetric (center-surround) structure of receptive fields in the retina.

The simplest models that attempt to capture both the receptive field properties and the response non-linearities are linear-nonlinear (LN) cascades, in which the incoming sensory stimulus is projected onto a linear kernel, and this linear response is then passed through a memoryless scalar nonlinear function whose output is used to generate the spiking response of the neuron. Such approaches have been used to make predictions about neural coding in general [7, 8], and, when combined with a constraint on the mean response level, to derive oriented receptive fields similar to those found in primary visual cortex [9, 10]. These models do not generally incorporate realistic levels of noise. And while the predictions are intuitively appealing, it is also somewhat of a mystery that they bypass the earlier (e.g., retinal) stages of visual processing, in which receptive fields are center-surround.

A number of authors have studied coding efficiency of scalar nonlinear functions in the presence of noise and compared them to neural responses to variables such as contrast [11, 12, 13, 14, 15]. Others have verified that the *distributions* of neural responses are in accordance with predictions of coding efficiency [16, 17, 18, 19]. To our knowledge, however, no previous result has attempted to jointly optimize the linear receptive field and the nonlinear response properties in the presence of realistic levels of input and output noise, and realistic constraints on response levels.

Here, we develop methods to optimize a full population of linear-nonlinear (LN) model neurons for transmitting information in natural images. We include a term in the objective function that captures metabolic costs associated with firing spikes [20, 21, 22]. We also include two sources of noise, in both input and output stages. We implement an algorithm for jointly optimizing the population of linear receptive fields and their associated nonlinearities. We find that, in the regime of significant noise, the optimal filters have a center-surround form, and the optimal nonlinearities are rectifying, consistent with response properties of retinal ganglion cells. We also observe asymmetries between the On- and the Off-center types similar to those measured in retinal populations. When both the input and the output noise are sufficiently small, our learning algorithm reduces to a generalized form of independent component analysis (ICA), yielding optimal filters that are localized and oriented, with corresponding smooth nonlinearities.

## 2 A model for noisy nonlinear efficient coding

We assume a neural model in the form of an LN cascade (Fig. 1a), which has been successfully fit to neural responses in retina, lateral geniculate nucleus, and primary visual cortex of primate visual systems [e.g., 23, 24, 25]. We develop a numerical method to optimize both the linear receptive fields and the corresponding point nonlinearities so as to maximize the information transmitted about natural images in the presence of input and output noise, as well as metabolic constraints on neural processing.

Consider a vector of inputs $\mathbf{x}$ of dimensionality $D$ (e.g. an image with $D$ pixels), and output vector $\mathbf{r}$ of dimensionality $J$ (the underlying firing rate of $J$ neurons). The response of a neuron $r_j$ is computed by taking an inner product of the (noise-corrupted) input with a linear filter $\mathbf{w}_j$ to obtain a generator signal $y_j$ (e.g. membrane voltage), which is then passed through neural nonlinearity $f_j$ (corresponding to the spike-generating process) and corrupted with additional neural noise,

$$r_j = f_j\left(y_j\right) + n_r \tag{1}$$

$$y_j = \mathbf{w}_j^T\left(\mathbf{x} + \mathbf{n}_x\right), \tag{2}$$

(Fig. 1a). Note that we did not constrain the model to be "complete" (the number of neurons can be smaller or larger than the input dimensionality) and that each neuron can have a different nonlinearity.

We aim to optimize an objective function that includes the mutual information between the input signal and the population responses, denoted $I(X; R)$, as well as an approximate measure of the metabolic operating cost of the system. It has been estimated that most of the energy expended by spiking neurons is associated with the cost of generating (and recovering from) spikes and that this cost is roughly proportional to the neural firing rate [22]. Thus we incorporate a penalty on the expected output, which gives the following objective function:

$$I(X; R) - \sum_j \lambda_j \left\langle r_j \right\rangle . \tag{3}$$

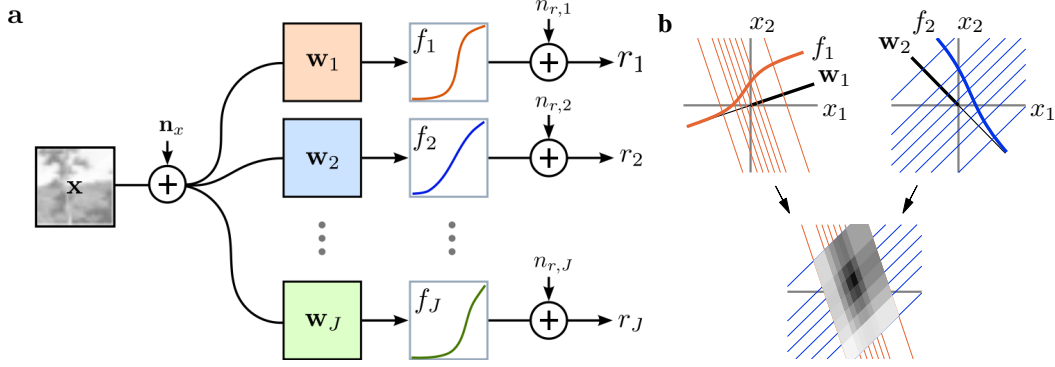

Figure 1: **a.** Schematic of the model (see text for description). The goal is to maximize information transfer between images $\mathbf{x}$ and the neural response $\mathbf{r}$, subject to metabolic cost of firing spikes. **b.** Information about the stimulus is conveyed both by the arrangement of the filters and the steepness of the neural nonlinearities. *Top:* two neurons encode two stimulus components (e.g. two pixels of an image, $x_1$ and $x_2$) with linear filters (black lines) whose output is passed through scalar nonlinear functions (thick color lines; thin color lines show isoresponse contours at evenly spaced output levels). The steepness of the nonlinearities specifies the precision with which each projection is represented: regions of steep slope correspond to finer partitioning of the input space, reducing the uncertainty about the input. *Bottom:* joint encoding leads to binning of the input space according to the isoresponse lines above. Grayscale shading indicates the level of uncertainty (entropy) in regions of the input (lighter shades correspond to higher uncertainty). Efficient codes optimize this binning, subject to input distribution, noise levels, and metabolic costs on the outputs.

Parameter $\lambda_j$ specifies the trade-off between information gained by firing more spikes, and the cost of generating them. It is difficult to obtain a biologically valid estimate for this parameter, and ultimately, the value of sensory information gained depends on the behavioral task and its context [26]. Alternatively, we can use $\lambda_j$ as a Lagrange multiplier to enforce the constraint on the mean output of each neuron.

Our goal is to adjust both the filters and the nonlinearities of the neural population so as to maximize the expectation of (3) under the joint distribution of inputs and outputs, $p(\mathbf{x}, \mathbf{r})$. We assume the filters are unit norm ($\|\mathbf{w}_j\| = 1$) to avoid an underdetermined model in which the nonlinearity scales along its input dimension to compensate for filter amplification. The nonlinearities $f_j$ are assumed to be monotonically increasing. We parameterized the *slope* of the nonlinearity $g_j = df_j/dy_j$ using a weighted sum of Gaussian kernels,

$$g_j(y_j|c_{jk}, \mu_{jk}, \sigma_j) = \sum_{k=1}^{K} c_{jk} \exp\left(-\frac{(y_j - \mu_{jk})^2}{2\sigma_j^2}\right) , \qquad (4)$$

with coefficients $c_{jk} \geq 0$. The number of kernels $K$ was chosen for sufficiently flexible nonlinearity (in our experiments $K = 500$). We spaced $\mu_{jk}$ evenly over the range of $y_j$ and chose $\sigma_j$ for smooth overlap of adjacent kernels (kernel centers $2\sigma_j$ apart).

## 2.1 Computing mutual information

How can we compute the information transmitted by the nonlinear network of neurons? Mutual information can be expressed as the difference between two entropies, $I(X; R) = H(X) - H(X|R)$. The first term is the entropy of the data, which is constant (i.e. it does not depend on the model) and can therefore be dropped from the objective function. The second term is the conditional differential entropy and represents the uncertainty in the input after observing the neural response. It is computed by taking the expectation over output values $H(X|R) = E_r\left[-\int p(\mathbf{x}|\mathbf{r}) \ln p(\mathbf{x}|\mathbf{r}) d\mathbf{x}\right]$. In general, computing the entropy of an arbitrary high dimensional distribution is not tractable. We make several assumptions that allow us to approximate the posterior, compute its entropy, and maximize mutual information. The posterior is proportional to the product of the likelihood and the prior, $p(\mathbf{x}|\mathbf{r}) \propto p(\mathbf{r}|\mathbf{x})p(\mathbf{x})$; below we describe these two functions in detail.

**The likelihood.** First, we assume the nonlinearity is smooth enough that, at the level of the noise (both input and output), $f_j$ can be linearized using first-order Taylor series expansion. This means that locally, for each input $\mathbf{x}^i$ and instance of noise,

$$\mathbf{r}^i \approx \mathbf{G}^i \mathbf{W}^T (\mathbf{x}^i + \mathbf{n}_x^i) + \mathbf{n}_r^i + \mathbf{f}_0^i, \tag{5}$$

where $\mathbf{W}$ is a matrix collecting the neural filters, $\mathbf{f}_0^i$ is a vector of constants, and $\mathbf{G}^i$ is a diagonal matrix containing the local derivatives of the response functions $g_j(y_j)$ at $y_j(\mathbf{x}^i)$. Here we have used $i$ to index parameters and random variables that change with each input. (Similar approximations have been used to minimize reconstruction error in neural nonlinearities [27] and maximize information in networks of interacting genes [28].)

If input and output noises are assumed to be constant and Gaussian, with covariances $\mathbf{C}_{n_x}$ and $\mathbf{C}_{n_r}$, respectively, we obtain a Gaussian likelihood $p(\mathbf{r}|\mathbf{x})$, with covariance

$$\mathbf{C}_{r|x}^i = \mathbf{G}^i \mathbf{W}^T \mathbf{C}_{n_x} \mathbf{W} \mathbf{G}^i + \mathbf{C}_{n_r}. \tag{6}$$

We emphasize that although the likelihood *locally* takes the form of a Gaussian distribution, its covariance is not fixed but depends on the input, leading to different values for the entropy of the posterior across the input space. Fig. 1b illustrates schematically how the organization of the filters and the nonlinearities affects the entropy and thus determines the precision with which neurons encode the inputs.

**The prior.** We would like to make as few assumptions as possible about the prior distribution of natural images. As described below, we rely on sampling image patches to approximate this density when computing $H(X|R)$. Nevertheless, to compute local estimates of the entropy we need to combine the prior with the likelihood. For smooth densities, the entropy depends on the curvature of the prior in the region where likelihood has significant mass. When an analytic form for the prior is available, we can use a second-order expansion of the prior around the maximum of the posterior (known as the "Laplace approximation" to the posterior). Unfortunately, this is difficult to compute reliably in high dimensions when only samples are available. Instead, we use the *global* curvature estimate in the form of the covariance matrix of the data, $\mathbf{C}_x$.

Putting these ingredients together, we compute the posterior as a product of two Gaussian distributions. This gives a Gaussian with covariance

$$\mathbf{C}_{x|r}^i = \left( \mathbf{C}_x^{-1} + \mathbf{W} \mathbf{G}^i (\mathbf{G}^i \mathbf{W}^T \mathbf{C}_{n_x} \mathbf{W} \mathbf{G}^i + \mathbf{C}_{n_r})^{-1} \mathbf{G}^i \mathbf{W}^T \right)^i \tag{7}$$

This provides a measure of uncertainty about each input and allows us to express information conveyed about the input ensemble by taking the expectation over the input and output distributions,

$$-H(X|R) = -E \left[ \frac{1}{2} \ln 2\pi e \det(\mathbf{C}_{x|r}^i) \right]. \tag{8}$$

We obtain Monte Carlo estimates of this conditional entropy by averaging the term in the brackets over a large ensemble of patches drawn from natural images and input/output noise sampled from assumed noise distributions.

## 2.2 Numerical optimization

We made updates to model parameters using online gradient ascent on the objective function computed on small batches of data. We omit the gradients here, as they are obtained using standard methods but do not yield easily interpretable update rules. One important special case is derived when the number of inputs equals the number of outputs, and both noise levels approach zero. In this setting, the update rule for the filters reduces to the ICA learning rule [8], with the gradient updates maximizing the entropy of the output distributions. Because our response constraint effectively limits the mean firing rate and not the maximum, the anti-Hebbian term is different from that found in standard ICA, and the optimal (maximum entropy) response distributions are exponential, rather than uniform. Note also that our method is more general than standard ICA: it adaptively adjusts the nonlinearities to match the input distribution, whereas standard ICA relies on a fixed nonlinear "contrast" function.

To ensure all nonlinearities were monotonically increasing, the coefficients $c_{jk}$ were adapted in log-space. After each step of gradient ascent, we normalized filters so that $\|\mathbf{w}_j\| = 1$. It was also

necessary to adjust the sampling of the nonlinearities (location of $\mu_{jk}$'s) because, as the fixed-norm filters rotated through input space, the variance of the projections can change drastically. Thus, whenever data fell outside the range, the range was doubled, and when all data fell inside the central 25%, it was halved.

# 3 Training the model on natural images

## 3.1 Methods

Natural image data were obtained by sampling $16 \times 16$ patches randomly from a collection of grayscale photographs of outdoor scenes [29], whose pixel intensities were linear w.r.t. light luminance levels. Importantly, we did not whiten images. The only preprocessing steps were to subtract the mean of each large image and rescale the image to attain a variance of 1 for the pixels.

We assumed that the input and output noises were i.i.d., so $\mathbf{C}_{n_x} = \sigma_{n_x}^2 \mathbf{I}_D$ and $\mathbf{C}_{n_r} = \sigma_{n_r}^2 \mathbf{I}_J$. We chose 8dB for the input ($\sigma_{n_x} \approx 0.4$). Although this is large relative to the variance of a pixel, as a result of strong spatial correlations in the input, some projections of the data (low frequency components) had SNR over 40dB. Output noise levels were set to -6dB (computed as $20 \log_{10}(\langle r_j \rangle / \sigma_{n_r})$; $\sigma_{n_r} = 2$) in order to match the high variability observed in retinal ganglion cells (see below). Parameter $\lambda_j$ was adjusted to attain an average rate of one spike per neuron per input image, $\langle r_j \rangle = 1$.

The model consisted of 100 neurons. We found this number to be sufficient to produce homogeneous sets of receptive fields that spatially tiled the image patch. In the retina, the ratio of inputs (cones) to outputs (retinal ganglion cells) varies greatly, from almost 1:3 in central fovea to more than 10:1 in the periphery [30]. Our ratio of 256:100 is within the physiological range, but other factors, such as eccentricity-dependent sampling, optical blur, and multiple ganglion cell subtypes make exact comparisons impossible.

We initialized filter weights and nonlinearity coefficients to random Gaussian values. Batch size was 100 patches, resampled after each update of the parameters. We trained the model for 100,000 iterations of gradient ascent with fixed step size. Initial conditions did not affect the learned parameters, with multiple runs yielding similar results. Unlike algorithms for training generative models, such as PCA or ICA, it is not possible to synthesize data from the LN model to verify convergence to the generating parameters.

## 3.2 Optimal filters and nonlinearities

We found that, in the presence of significant input and output noise, the optimal filters have center-surround structure, rather than the previously reported oriented shapes (Fig. 2a). Neurons organize into two populations with On-center and Off-center filters, each independently tiling the visual space. The population contains fewer On-center neurons (41 of 100) and their filters are spatially larger (Fig. 2b). These results are consistent with measurements of receptive field structure in retinal ganglion cells [31] (Fig. 3).

The optimal nonlinear functions show hard rectification, with thresholds near the mode of the input distribution (Fig. 2c). Measured neural nonlinearities are typically softer, but when rectified noise is taken into account, a hard-rectified model has been shown to be a good description of neural variability [32]. The combination of hard-rectifying nonlinearities and On/Off filter organization means that the subspace encoded by model neurons is approximately half the dimensionality of the output. For substantial levels of noise, we find that even a "complete" network (in which the number of outputs equals the number of inputs) does not span the input space and instead encodes the subspace with highest signal power.

The metabolic cost parameters $\lambda_j$ that yielded the target output rate were close to 0.2. This means that increasing the firing rate of each neuron by one spike per image leads to an information gain of 20 bits for the entire population. This value is consistent with previous estimates of 40-70 bits per second for the optic nerve [33], and an assumption of 2-5 fixations (and thus unique images seen) per second.

To examine the effect of noise on optimal representations, we trained the model under different regimes of noise (Fig. 4). We found that decreasing input noise leads to smaller filters and a reduction

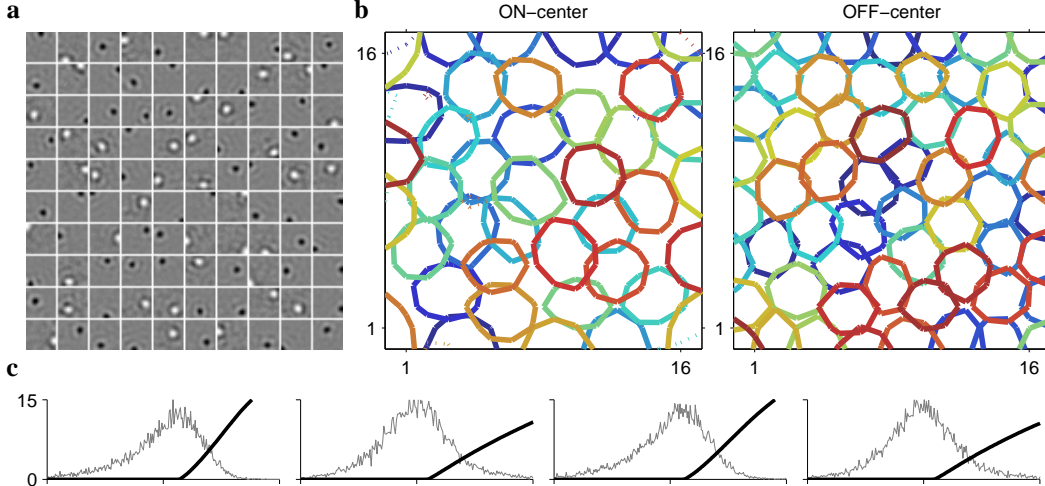

Figure 2: In the presence of biologically realistic level of noise, the optimal filters are center-surround and contain both On-center and Off-center profiles; the optimal nonlinearities are hard-rectifying functions. **a**. The set of learned filters for 100 model neurons. **b**. In pixel coordinates, contours of On-center (Off-center) filters at 50% maximum (minimum) levels. **c**. The learned non-linearities for the first four model neurons, superimposed on distributions of filter outputs.

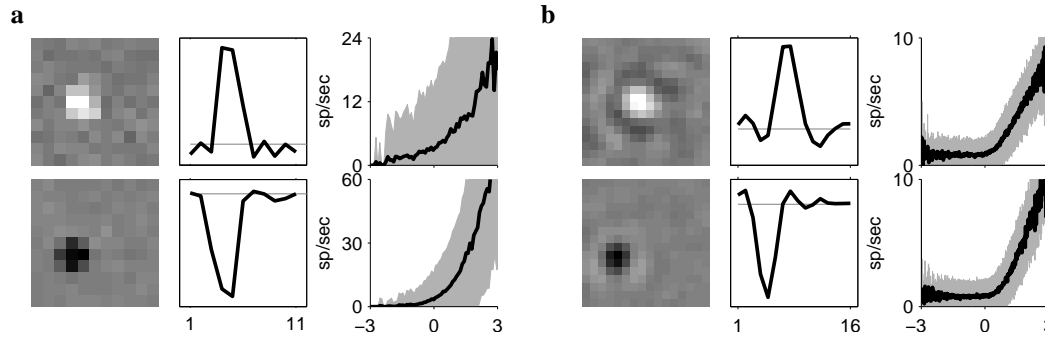

Figure 3: **a**. A characterization of two retinal ganglion cells obtained with white noise stimulus [31]. We plot the estimated linear filters, horizontal slices through the filters, and mean output as a function of input (black line, shaded area shows one standard deviation of response). **b**. For comparison, we performed the same analysis on two model neurons. Note that the spatial scales of model and data filters are different.

in the number of On-center neurons (bottom left panel). In this case, increasing the number of neurons restored the balance of On- and Off-center filters (not shown). In the case of vanishing input and output noise, we obtain localized oriented filters (top left panel), and the nonlinearities are smoothly accelerating functions that map inputs to an exponential output distribution (not shown). These results are consistent with previous theoretical work showing that optimal nonlinearity in the low noise regime maximizes the entropy of the output subject to response constraints [11, 7, 17].

How important is the choice of linear filters for efficient information transmission? We compared the performance of different filtersets across a range of firing rates (Fig. 5). For each simulation, we re-optimized the nonlinearities, adjusting $\lambda_j$'s for desired mean rate, while holding the filters fixed. As a rough estimate of input entropy $H(X)$, we used an upper bound – a Gaussian distribution with the covariance of natural images. Our results show that when filters are mismatched to the noise levels, performance is significantly degraded. At equivalent output rate, the "wrong" filters transmit approximately 10 fewer bits; conversely, it takes about 50% more spikes to encode the same amount of information.

We also compared the coding efficiency of networks with variable number of neurons. First, we fixed the allotted population spike budget to 100 (per input), fixed the absolute output noise, and

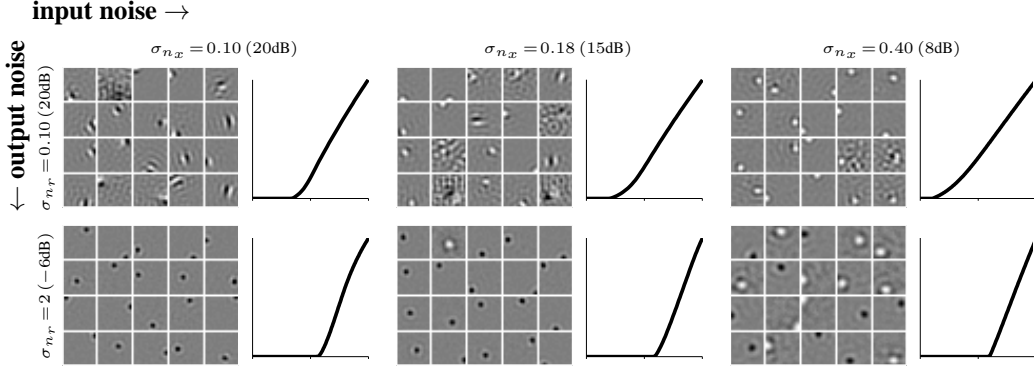

Figure 4: Each panel shows a subset of filters (20 of 100) obtained under different levels of input and output noise, as well as the nonlinearity for a typical neuron in each model.

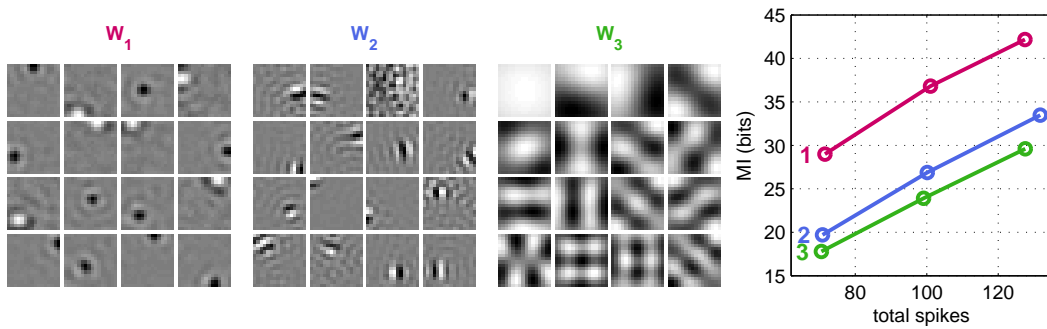

Figure 5: Information transmitted as a function of spike rate, under noisy conditions (8dB $SNR_{in}$, $-6$dB $SNR_{out}$). We compare the performance of optimal filters ($\mathbf{W}_1$) to filters obtained under low noise conditions ($\mathbf{W}_2$, 20dB $SNR_{in}$, 20dB $SNR_{out}$) and PCA filters, i.e. the first 100 eigenvectors of the data covariance matrix ($\mathbf{W}_3$).

varied the number of neurons from 1 (very precise) neuron to 150 (fairly noisy) neurons (Fig. 6a). We estimated the transmitted information as described above. In this regime of noise and spiking budget, the optimal population size was around 100 neurons. Next, we repeated the analysis but used neurons with fixed precision, i.e., the spike budget was scaled with the population to give 1 noisy neuron or 150 equally noisy neurons (Fig. 6b). As the population grows, more information is transmitted, but the rate of increase slows. This suggests that incorporating an additional penalty, such as a fixed metabolic cost per neuron, would allow us to predict the optimal number of canonical noisy neurons.

## 4   Discussion

We have described an efficient coding model that incorporates ingredients essential for computation in sensory systems: non-Gaussian signal distributions, realistic levels of input and output noise, metabolic costs, nonlinear responses, and a large population of neurons. The resulting optimal solution mimics neural behaviors observed in the retina: a combination of On and Off center-surround receptive fields, halfwave-rectified nonlinear responses, and pronounced asymmetries between the On- and the Off- populations. In the noiseless case, our method provides a generalization of ICA and produces localized, oriented filters.

In order to make the computation of entropy tractable, we made several assumptions. First, we assumed a smooth response nonlinearity, to allow local linearization when computing entropy. Although some of our results produce non-smooth nonlinearities, we think it unlikely that this systematically affected our findings; nevertheless, it might be possible to obtain better estimates by considering higher order terms of local Taylor expansion. Second, we used the global curvature of the prior density to estimate the local posterior in Eqn. 7. A better approximation would be obtained

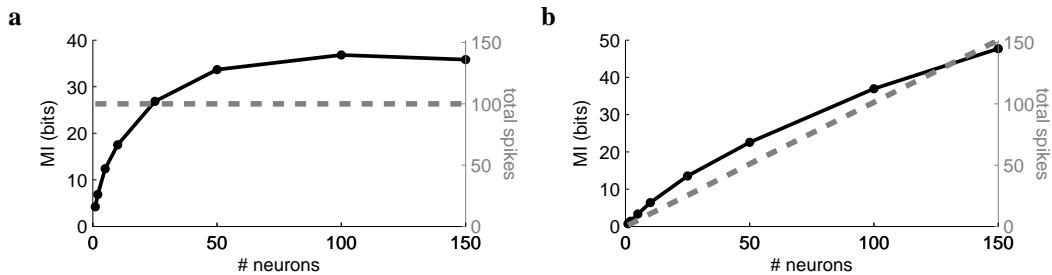

Figure 6: Transmitted information (solid line) and total spike rate (dashed line) as a function of the number of neurons, assuming (**a**) fixed *total* spike budget and (**b**) fixed spike budget *per neuron*.

from an adaptive second-order expansion of the prior density around the maximum of the posterior. This requires the estimation of local density (or rather, its curvature) from samples, which is a non-trivial problem in a high-dimensional space.

Our results bear some resemblance to previous attempts to derive retinal properties as optimal solutions. Most notably, optimal linear transforms that optimize information transmission under a constraint on total response power have been shown to be consistent with center-surround [4] and more detailed [34] shapes of retinal receptive fields. But such linear models do not provide a unique solution, nor can they make predictions about nonlinear behaviors. An alternative formulation, using linear basis functions to *reconstruct* the input signal, has also been shown to exhibit center-surround shapes [35, 6]. However, this approach makes additional assumptions about the sparsity of weights in linear filters, nor does it explicitly maximize the efficiency of the code.

Our results suggest several directions for future efforts. First, noise in our model is a known constant value. In contrast, neural systems must deal with changing levels of noise and signal, and must estimate them based only on their inputs. An interesting question, unaddressed in current work, is how to adapt representations (e.g., synaptic weights and nonlinearities) to dynamically regulate coding efficiency. Second, we are interested in extending this model to make predictions about higher visual areas. We do not interpret our results in the noiseless case (oriented, localized filters) as predictions for optimal cortical representations. Instead, we intend to extend this framework to cortical representations that must deal with accumulated nonlinearity and noise arising from previous stages of the processing hierarchy.

# References

[1] F. Attneave, "Some informational aspects of visual perception.," *Psychological Review*, vol. 61, no. 3, pp. 183–193, 1954.

[2] H. Barlow, "Possible principles underlying the transformations of sensory messages," in *Sensory Communication*, pp. 217–234, MIT Press, 1961.

[3] M. V. Srinivasan, S. B. Laughlin, and A. Dubs, "Predictive coding: A fresh view of inhibition in the retina," *Proceedings of the Royal Society of London. Series B. Biological Sciences*, vol. 216, pp. 427 –459, Nov. 1982.

[4] J. J. Atick and A. N. Redlich, "Towards a theory of early visual processing," *Neural Computation*, vol. 2, no. 3, pp. 308–320, 1990.

[5] J. J. Atick, "Could information theory provide an ecological theory of sensory processing?," *Network Computation in Neural Systems*, vol. 3, no. 2, pp. 213–251, 1992.

[6] E. Doi and M. S. Lewicki, "A theory of retinal population coding," in *Advances in Neural Information Processing Systems 19* (B. Schölkopf, J. Platt, and T. Hoffman, eds.), pp. 353–360, Cambridge, MA: MIT Press, 2007.

[7] J. Nadal and N. Parga, "Nonlinear neurons in the low-noise limit: a factorial code maximizes information transfer," *Network: Computation in Neural Systems*, vol. 5, no. 4, pp. 565–581, 1994.

[8] A. J. Bell and T. J. Sejnowski, "An Information-Maximization approach to blind separation and blind deconvolution," *Neural Computation*, vol. 7, no. 6, pp. 1129–1159, 1995.

[9] B. A. Olshausen and D. J. Field, "Emergence of simple-cell receptive field properties by learning a sparse code for natural images," *Nature*, vol. 381, no. 6583, pp. 607–609, 1996.

[10] A. J. Bell and T. J. Sejnowski, "The "independent components" of natural scenes are edge filters," *Vision Research*, vol. 37, no. 23, pp. 3327–3338, 1997.

[11] S. Laughlin, "A simple coding procedure enhances a neuron's information capacity," *Z Naturforsch*, no. Sep-Oct, 1981.

[12] A. Treves, S. Panzeri, E. T. Rolls, M. Booth, and E. A. Wakeman, "Firing rate distributions and efficiency of information transmission of inferior temporal cortex neurons to natural visual stimuli," *Neural Computation*, vol. 11, no. 3, p. 601–631, 1999.

[13] N. Brenner, W. Bialek, and R. de Ruyter van Steveninck, "Adaptive rescaling maximizes information transmission," *Neuron*, vol. 26, no. 3, pp. 695–702, 2000. PMID: 10896164.

[14] A. L. Fairhall, G. D. Lewen, W. Bialek, and R. R. de Ruyter van Steveninck, "Efficiency and ambiguity in an adaptive neural code," *Nature*, vol. 412, no. 6849, p. 787–792, 2001.

[15] M. D. McDonnell and N. G. Stocks, "Maximally informative stimuli and tuning curves for sigmoidal Rate-Coding neurons and populations," *Physical Review Letters*, vol. 101, no. 5, p. 058103, 2008.

[16] W. B. Levy and R. A. Baxter, "Energy efficient neural codes," *Neural Computation*, vol. 8, no. 3, pp. 531–543, 1996.

[17] R. Baddeley, L. F. Abbott, M. C. Booth, F. Sengpiel, T. Freeman, E. A. Wakeman, and E. T. Rolls, "Responses of neurons in primary and inferior temporal visual cortices to natural scenes.," *Proceedings of the Royal Society B: Biological Sciences*, vol. 264, no. 1389, pp. 1775–1783, 1997.

[18] V. Balasubramanian and M. J. Berry, "A test of metabolically efficient coding in the retina," *Network: Computation in Neural Systems*, vol. 13, no. 4, p. 531–552, 2002.

[19] L. Franco, E. T. Rolls, N. C. Aggelopoulos, and J. M. Jerez, "Neuronal selectivity, population sparseness, and ergodicity in the inferior temporal visual cortex," *Biol. Cybernetics*, vol. 96, no. 6, pp. 547–560, 2007.

[20] S. B. Laughlin, R. R. V. Steveninck, and J. C. Anderson, "The metabolic cost of neural information," *Nat. Neurosci*, vol. 1, no. 1, p. 36–41, 1998.

[21] P. Lennie, "The cost of cortical computation," *Current Biology*, vol. 13, pp. 493–497, Mar. 2003.

[22] D. Attwell and S. B. Laughlin, "An energy budget for signaling in the grey matter of the brain," *Journal of Cerebral Blood Flow and Metabolism*, vol. 21, no. 10, pp. 1133–1145, 2001.

[23] D. K. Warland, P. Reinagel, and M. Meister, "Decoding visual information from a population of retinal ganglion cells," *Journal of Neurophysiology*, vol. 78, no. 5, pp. 2336 –2350, 1997.

[24] J. W. Pillow, L. Paninski, V. J. Uzzell, E. P. Simoncelli, and E. J. Chichilnisky, "Prediction and decoding of retinal ganglion cell responses with a probabilistic spiking model," *The Journal of Neuroscience*, vol. 25, no. 47, pp. 11003 –11013, 2005.

[25] D. J. Heeger, "Half-Squaring in responses of cat striate cells," *Visual Neuroscience*, vol. 9, no. 05, pp. 427–443, 1992.

[26] W. Bialek, R. van Steveninck, and N. Tishby, "Efficient representation as a design principle for neural coding and computation," in *IEEE International Symposium on Information Theory*, pp. 659–663, 2006.

[27] T. von der Twer and D. I. A. MacLeod, "Optimal nonlinear codes for the perception of natural colours," *Network: Computation in Neural Systems*, vol. 12, no. 3, pp. 395–407, 2001.

[28] A. M. Walczak, G. Tkačik, and W. Bialek, "Optimizing information flow in small genetic networks. II. feed-forward interactions," *Physical Review E*, vol. 81, no. 4, p. 041905, 2010.

[29] E. Doi, T. Inui, T.-W. Lee, T. Wachtler, and T. J. Sejnowski, "Spatiochromatic receptive field properties derived from information-theoretic analyses of cone mosaic responses to natural scenes," *Neural Computation*, vol. 15, pp. 397–417, 2003.

[30] H. Wässle, U. Grünert, J. Röhrenbeck, and B. B. Boycott, "Retinal ganglion cell density and cortical magnification factor in the primate," *Vision Research*, vol. 30, no. 11, pp. 1897–1911, 1990.

[31] E. J. Chichilnisky and R. S. Kalmar, "Functional asymmetries in ON and OFF ganglion cells of primate retina," *The Journal of Neuroscience*, vol. 22, no. 7, pp. 2737 –2747, 2002.

[32] M. Carandini, "Amplification of Trial-to-Trial response variability by neurons in visual cortex," *PLoS Biol*, vol. 2, no. 9, p. e264, 2004.

[33] C. L. Passaglia and J. B. Troy, "Information transmission rates of cat retinal ganglion cells," *Journal of Neurophysiology*, vol. 91, no. 3, pp. 1217 –1229, 2004.

[34] E. Doi, J. L. Gauthier, G. D. Field, J. Shlens, A. Sher, M. Greschner, T. Machado, K. Mathieson, D. Gunning, A. M. Litke, L. Paninski, E. J. Chichilnisky, and E. P. Simoncelli, "Redundant representations in macaque retinal populations are consistent with efficient coding," in *Computational and Systems Neuroscience (CoSyNe)*, February 2011.

[35] B. T. Vincent and R. J. Baddeley, "Synaptic energy efficiency in retinal processing," *Vision Research*, vol. 43, no. 11, pp. 1285–1292, 2003.

